# Sample Propagation

**Mark A. Paskin**
Computer Science Division
University of California, Berkeley
Berkeley, CA 94720
mark@paskin.org

## Abstract

Rao–Blackwellization is an approximation technique for probabilistic inference that flexibly combines exact inference with sampling. It is useful in models where conditioning on some of the variables leaves a simpler inference problem that can be solved tractably. This paper presents *Sample Propagation*, an efficient implementation of Rao–Blackwellized approximate inference for a large class of models. Sample Propagation tightly integrates sampling with message passing in a junction tree, and is named for its simple, appealing structure: it walks the clusters of a junction tree, sampling some of the current cluster's variables and then passing a message to one of its neighbors. We discuss the application of Sample Propagation to conditional Gaussian inference problems such as switching linear dynamical systems.

## 1 Introduction

Message passing on junction trees is an efficient means of solving many probabilistic inference problems [1, 2]. However, as these are exact methods, their computational costs must scale with the complexity of the inference problem, making them inapplicable to very demanding inference tasks. This happens when the messages become too expensive to compute, as in discrete models of large treewidth or conditional Gaussian models [3].

In these settings it is natural to investigate whether junction tree techniques can be combined with sampling to yield fast, accurate approximate inference algorithms. One way to do this is to use sampling to approximate the messages, as in HUGS [4, 5]. This strategy has two disadvantages: first, the samples must be stored, which limits the sample size by space constraints (rather than time constraints); and second, variables are sampled using only local information, leading to samples that may not be likely under the entire model.

Another way to integrate sampling and message passing is via Rao–Blackwellization, where we repeatedly sample a subset of the model's variables and then compute all of the messages exactly, conditioned on these sample values. This technique, suggested in [6] and studied in [7], yields a powerful and flexible approximate inference algorithm; however, it can be expensive because the junction tree algorithm must be run for every sample.

In this paper, we present a simple implementation of Rao–Blackwellized approximate inference that avoids running the entire junction tree algorithm for every sample. We develop a new message passing algorithm for junction trees that supports fast retraction of evidence,

and we tightly integrate it with a blocking Gibbs sampler so that only one message must be recomputed per sample. The resulting algorithm, *Sample Propagation*, has an appealing structure: it walks the clusters of a junction tree, resampling some of the current cluster's variables and then passing a message to the next cluster in the walk.

## 2 Rao–Blackwellized approximation using junction tree inference

We start by presenting our notation and assumptions on the probability model. Then we summarize the three basic ingredients of our approach: message passing in a junction tree, Rao–Blackwellized approximation, and sampling via Markov chain Monte Carlo.

### 2.1 The probability model

Let $X = (X_i : i \in I)$ be a vector of random variables indexed by the finite, ordered set $I$, and for each index $i$ let $\mathcal{X}_i$ be the range of $X_i$. We will use the symbols $A, B, C, D$ and $E$ to denote subsets of the index set $I$. For each subset $A$, let $X_A \equiv (X_i : i \in A)$ be the corresponding subvector of random variables and let $\mathcal{X}_A$ be its range.

It greatly simplifies the exposition to develop a simple notation for assignments of values to subsets of variables. An *assignment* to a subset $A$ is a set of pairs $\{(i, x_i) : i \in A\}$, one per index $i \in A$, where $x_i \in \mathcal{X}_i$. We use the symbols $\mathbf{u}, \mathbf{v}$, and $\mathbf{w}$ to represent assignments, and we use $\mathbb{X}_A$ to denote the set of assignments to the subset $A$ (with the shorthand $\mathbb{X} \equiv \mathbb{X}_I$).

We use two operations to generate new assignments from old assignments. Given assignments $\mathbf{u}$ and $\mathbf{v}$ to disjoint subsets $A$ and $B$, respectively, their union $\mathbf{u} \cup \mathbf{v}$ is an assignment to $A \cup B$. If $\mathbf{u}$ is an assignment to $A$ then the *restriction* of $\mathbf{u}$ to another subset $B$ is $\mathbf{u}_B \equiv \{(i, x_i) \in \mathbf{u} : i \in B\}$, an assignment to $A \cap B$. We also let functions act on assignments in the natural way: if $\mathbf{u} = \{(i, x_i) : i \in D\}$ is an assignment to $D$ and $f$ is a function whose domain is $\mathcal{X}_D$, then we use $f(\mathbf{u})$ to denote $f(x_i : i \in D)$.

We consider probability densities of the form

$$p(\mathbf{u}) \propto \prod_{C \in \mathbf{C}} \psi_C(\mathbf{u}_C), \qquad \mathbf{u} \in \mathbb{X} \tag{1}$$

where $\mathbf{C}$ is a set of subsets of $I$ and each $\psi_C$ is a potential function over $C$ (i.e., a non-negative function of $\mathcal{X}_C$). This class includes directed graphical models (i.e., Bayesian networks) and undirected graphical models such as Markov random fields. Observed variables are reflected in the model by evidence potentials. We use $p_A(\cdot)$ to denote the marginal density of $X_A$ and $p_{A|B}(\cdot \mid \cdot)$ to denote the conditional density of $X_A$ given $X_B$. Finally, we use the notation of finite measure spaces for simplicity, but our approach extends to the continuous case.

### 2.2 Junction tree inference

Given a density of the form (1), we view the problem of probabilistic inference as that of computing the expectation of a function $f$: $\mathbb{E}\left[f(X)\right] = \sum_{\mathbf{u} \in \mathbb{X}} p(\mathbf{u}) f(\mathbf{u})$. This sum can be expensive to compute when $\mathbb{X}$ is a large space. When the desired expectation is "local" in that $f$ depends only upon some subset of the variables $X_D$, we can compute the expectation more cheaply using a marginal density as

$$\mathbb{E}\left[f(X_D)\right] = \sum_{\mathbf{u} \in \mathbb{X}_C} p_C(\mathbf{u}) f(\mathbf{u}_D) \tag{2}$$

where $p_C$ is the marginal density of $X_C$ and $C \supseteq D$ "covers" the input of the function. If this sum is tractable, then we have reduced the problem to that of computing $p_C$.

We can compute this marginal via message passing on a junction tree [1, 2]. A *junction tree* for $\mathbf{C}$ is a singly-connected, undirected graph $(\mathbf{C}, \mathbf{E})$ with the *junction tree property*: for each pair of nodes (or *clusters*) $A, B \in \mathbf{C}$ that contain some $i \in I$, every cluster on

the unique path between $A$ and $B$ also contains $i$. In what follows we assume we have a junction tree for $\mathbf{C}$ with a cluster that covers $D$, the input of $f$. (Such a junction tree can always be found, but we may have to enlarge the subsets in $\mathbf{C}$.)

Whereas the HUGIN message passing algorithm [2] may be more familiar, Sample Propagation is most easily described by extending the Shafer–Shenoy algorithm [1]. In this algorithm, we define for each edge $B \to C$ of the junction tree a potential over $B \cap C$:

$$\mu_{BC}(\mathbf{u}) \equiv \sum_{\mathbf{v} \in \mathbb{X}_{B \setminus C}} \psi_B(\mathbf{u} \cup \mathbf{v}) \prod_{\substack{(A,B) \in \mathbf{E} \\ A \neq C}} \mu_{AB}(\mathbf{u}_A \cup \mathbf{v}_A), \qquad \mathbf{u} \in \mathbb{X}_{B \cap C} \tag{3}$$

$\mu_{BC}$ is called the *message from $B$ to $C$*. Note that this definition is recursive—messages can depend on each other—with the base case being messages from leaf clusters of the junction tree. For each cluster $C$ we define a potential $\beta_C$ over $C$ by

$$\beta_C(\mathbf{u}) \equiv \psi_C(\mathbf{u}) \prod_{(B,C) \in \mathbf{E}} \mu_{BC}(\mathbf{u}_B), \qquad \mathbf{u} \in \mathbb{X}_C \tag{4}$$

$\beta_C$ is called the *cluster belief of $C$*, and it follows that $\beta_C \propto p_C$, i.e., that the cluster beliefs are the marginals over their respective variables (up to renormalization). Thus we can use the (normalized) cluster beliefs $\beta_C$ for some $C \supseteq D$ to compute the expectation (2).

In what follows we will also be interested in computing conditional cluster densities given an evidence assignment $\mathbf{w}$ to a subset of the variables $X_E$. Because $p_{I \setminus E|E}(\mathbf{u} \,|\, \mathbf{w}) \propto p(\mathbf{u} \cup \mathbf{w})$, we can "enter in" this evidence by instantiating $\mathbf{w}$ in every cluster potential $\psi_C$. The cluster beliefs (4) will then be proportional to the conditional density $p_{C \setminus E|E}(\cdot \,|\, \mathbf{w})$.

Junction tree inference is often the most efficient means of computing exact solutions to inference problems of the sort described above. However, the sums required by the messages (3) or the function expectations (2) are often prohibitively expensive to compute. If the variables are all finite-valued, this happens when the clusters of the junction tree are too large; if the model is conditional-Gaussian, this happens when the messages, which are mixtures of Gaussians, have too many mixture components [3].

**2.3 Rao–Blackwellized approximate inference**

In cases where the expectation is intractable to compute exactly, it can be approximated by a *Monte Carlo estimate*:

$$\mathbb{E}\left[f(X_D)\right] \approx \frac{1}{N} \sum_{n=1}^{N} f(\mathbf{v}_D^n) \tag{5}$$

where $\{\mathbf{v}^n : 1 \leq n \leq N\}$ are a set of samples of $X$. However, obtaining a good estimate will require many samples if $f(X_D)$ has high variance.

Many models have the property that while computing exact expectations is intractable, there exists a subset of random variables $X_E$ such that the conditional expectation $\mathbb{E}\left[f(X_D) \,|\, X_E = x_E\right]$ can be computed efficiently. This leads to the *Rao–Blackwellized estimate*, where we use a set of samples $\{\mathbf{w}^n : 1 \leq n \leq N\}$ of $X_E$ to approximate

$$\mathbb{E}\left[f(X_D)\right] = \mathbb{E}\left[\mathbb{E}\left[f(X_D) \,|\, X_E\right]\right] \approx \frac{1}{N} \sum_{n=1}^{N} \mathbb{E}\left[f(X_D) \,|\, \mathbf{w}^n\right] \tag{6}$$

The first advantage of this scheme over standard Monte Carlo integration is that the Rao–Blackwell theorem guarantees that the expected squared error of the estimate (6) is upper bounded by that of (5), and strictly so when $f(X_D)$ depends on $X_{D \setminus E}$. A second advantage is that (6) requires samples from a smaller (and perhaps better-behaved) probability space.

---
**Algorithm 1** Rao–Blackwell estimation on a junction tree
---
**Input:** A set of samples $\{\mathbf{w}^n : 1 \le n \le N\}$ of $X_E$, a function $f$ of $\mathcal{X}_D$, and a cluster $C \supseteq D$
**Output:** An estimate $\hat{f} \approx \mathbb{E}[f(X_D)]$
 1: Initialize the estimator $\hat{f} = 0$.
 2: **for** $n = 1$ to $N$ **do**
 3:    Enter the assignment $\mathbf{w}^n$ as evidence into the junction tree.
 4:    Use message passing to compute the beliefs $\beta_C \propto p_{C \setminus E|E}(\cdot \mid \mathbf{w}^n)$ via (3) and (4).
 5:    Compute the expectation $\mathbb{E}[f(X_D) \mid \mathbf{w}^n]$ via (7).
 6:    Set $\hat{f} = \hat{f} + \mathbb{E}[f(X_D) \mid \mathbf{w}^n]$.
 7: Set $\hat{f} = \hat{f}/N$.
---

However, the Rao–Blackwellized estimate (6) is more expensive to compute than (5) because we must compute conditional expectations. In many cases, message passing in a junction tree can be used to implement these computations (see Algorithm 1). We can enter each sample assignment $\mathbf{w}^n$ as evidence into the junction tree and use message passing to compute the conditional density $p_{C \setminus E|E}(\cdot \mid \mathbf{w}^n)$ for some cluster $C$ that covers $D$. We then compute the conditional expectation as

$$\mathbb{E}[f(X_D) \mid \mathbf{w}^n] = \sum_{\mathbf{u} \in \mathbb{X}_{C \setminus E}} p_{C \setminus E|E}(\mathbf{u} \mid \mathbf{w}^n) f(\mathbf{u}_D \cup \mathbf{w}_D^n) \tag{7}$$

### 2.4 Markov chain Monte Carlo

We now turn to the problem of obtaining the samples $\{\mathbf{w}^n\}$ of $X_E$. Markov chain Monte Carlo (MCMC) is a powerful technique for generating samples from a complex distribution $p$; we design a Markov chain whose stationary distribution is $p$, and simulate the chain to obtain samples [8]. One simple MCMC algorithm is the Gibbs sampler, where each successive state of the Markov chain is chosen by resampling one variable conditioned on the current values of the remaining variables. A more advanced technique is "blocking" Gibbs sampling, where we resample a subset of variables in each step; this technique can yield Markov chains that mix more quickly [9].

To obtain the benefits of sampling in a smaller space, we would like to sample directly from the marginal $p_E$; however, this requires us to sum out the nuisance variables $X_{I \setminus E}$ from the joint density $p$. Blocking Gibbs sampling is particularly attractive in this setting because message passing can be used to implement the required marginalizations.[1] Assume that the current state of the Markov chain over $X_E$ is $\mathbf{w}^n$. To generate the next state of the chain $\mathbf{w}^{n+1}$ we choose a cluster $C$ (randomly, or according to a schedule) and resample $X_{C \cap E}$ given $\mathbf{w}_{E \setminus C}^n$; i.e., we resample the $E$ variables within $C$ given the $E$ variables outside $C$. The transition density can be computed by entering the evidence $\mathbf{w}_{E \setminus C}^n$ into the junction tree, computing the cluster belief at $C$, and marginalizing down to a conditional density over $X_{C \cap E}$. The complete Gibbs sampler is given as Algorithm 2.[2]

## 3 Sample Propagation

Algorithms 1 and 2 represent two of the three key ideas behind our proposal: both Gibbs sampling and Rao–Blackwellized estimation can be implemented efficiently using message passing on a junction tree. The third idea is that these two uses of message passing can be interleaved so that each sample requires only one message to be computed.

**Algorithm 2** Blocking Gibbs sampler on a junction tree

---

**Input:** A subset of variables $X_E$ to sample and a sample size $N$
**Output:** A set of samples $\{\mathbf{w}^n : 1 \leq n \leq N\}$ of $X_E$
 1: Choose an initial assignment $\mathbf{w}^0 \in \mathbb{X}_E$.
 2: **for** $n = 1$ to $N$ **do**
 3:     Choose a cluster $C \in \mathbf{C}$.
 4:     Enter the evidence $\mathbf{w}_{E \setminus C}^{n-1}$ into the junction tree.
 5:     Use message passing to compute the beliefs $\beta_C \propto p_{C|E \setminus C}(\cdot \,|\, \mathbf{w}_{E \setminus C}^{n-1})$ via (3) and (4).
 6:     Marginalize over $X_{C \setminus E}$ to obtain the transition density $p_{C \cap E|E \setminus C}(\cdot \,|\, \mathbf{w}_{E \setminus C}^{n-1})$.
 7:     Sample $\mathbf{w}_{C \cap E}^n \sim p_{C \cap E|E \setminus C}(\cdot \,|\, \mathbf{w}_{E \setminus C}^{n-1})$ and set $\mathbf{w}_{E \setminus C}^n = \mathbf{w}_{E \setminus C}^{n-1}$.

---

## 3.1 Lazy updating of the Rao–Blackwellized estimates

Algorithms 1 and 2 both process the samples sequentially, so the first advantage of merging them is that the sample set need not be stored. The second advantage is that, by being selective about when the Rao–Blackwellized estimator is updated, we can compute the messages once, not twice, per sample.

When the Gibbs sampler chooses to resample a cluster $C$ that covers $D$ (the input of $f$), we can update the Rao–Blackwellized estimator for free. In particular, the Gibbs sampler computes the cluster belief $\beta_C \propto p_{C|E \setminus C}(\cdot \,|\, \mathbf{w}_{E \setminus C}^{n-1})$ in order to compute the transition density $p_{C \cap E|E \setminus C}(\cdot \,|\, \mathbf{w}_{E \setminus C}^{n-1})$. Once it samples $\mathbf{w}_{C \cap E}^n$ from this density, we can instantiate the sample in the belief $\beta_C$ to obtain the conditional density $p_{C \setminus E|E}(\cdot \,|\, \mathbf{w}^n)$ needed by the Rao–Blackwellized estimator. (This follows from the fact that $\mathbf{w}_{E \setminus C}^n = \mathbf{w}_{E \setminus C}^{n-1}$.) In fact, when it is tractable to do so, we can simply use the cluster belief $\beta_C$ to update the estimator in (7); because it treats more variables exactly, it can yield a lower-variance estimate.

Therefore, if we are willing to update the Rao–Blackwellized estimator only when the Gibbs sampler chooses a cluster that covers the function's inputs, we can focus on reducing the computational requirements of the Gibbs sampler. In this scheme the estimate will be based on fewer samples, but the samples that are used will be less correlated because they are more distant from each other in the Markov chain. In parallel estimation problems where every cluster is computing expectations, every sample will be used to update an estimate, but not every estimate will be updated by every sample.

## 3.2 Optimizing the Gibbs sampler

We now turn to the Gibbs sampler. The Gibbs sampler computes the messages so that it can compute the cluster belief $\beta_C$ when it resamples within a cluster $C$. An important property of the computation (4) is that it requires only those messages directed towards $C$; thus, we have again reduced by half the number of messages required per sample.

The difficulty in further minimizing the number of messages computed by the Gibbs sampler is that the evidence on the junction tree is constantly changing. It will therefore be useful to modify the message passing so that, rather than instantiating all the evidence and then passing messages, the evidence is instantiated on the fly, on a per-cluster basis. For each edge $B \to C$ we define a potential $\mu_{BC|E}$ by

$$\mu_{BC|E}(\mathbf{u}, \mathbf{w}) \equiv \sum_{\mathbf{v} \in \mathbb{X}_{B \setminus (C \cup E)}} \psi_B(\mathbf{u} \cup \mathbf{v} \cup \mathbf{w}_{B \setminus C}) \prod_{\substack{A \neq C \\ (A,B) \in \mathbf{E}}} \mu_{AB|E}((\mathbf{u} \cup \mathbf{v} \cup \mathbf{w}_{B \setminus C})_A, \mathbf{w}) \quad (8)$$

where $\mathbf{u} \in \mathbb{X}_{B \cap C}$ and $\mathbf{w} \in \mathbb{X}_E$. This is the *conditional message from $B$ to $C$ given evidence $\mathbf{w}$ on $X_E$*. Figure 1 illustrates how the ranges of the assignment variables $\mathbf{u}$, $\mathbf{v}$, and $\mathbf{w}$ cover the variables of $B$; the intuition is that when we send a message from $B$ to $C$, we instantiate all evidence variables that are in $B$ but not those that are in $C$; this gives us

**Algorithm 3** Sample Propagation

**Input:** A function $f$ of $\mathcal{X}_D$, a cluster $C \supseteq D$, a subset $E$ to sample, and a sample size $N$
**Output:** An estimate $\hat{f} \approx \mathbb{E}\left[f(X_D)\right]$
1: Choose an initial assignment $\mathbf{w}^0 \in \mathbb{X}_E$ and compute the messages $\mu_{AB|E}(\cdot, \mathbf{w}^0)$ via (8).
2: Choose a cluster $C_1 \in \mathbf{C}$, initialize the estimator $\hat{f} = 0$, and set the sample count $M = 0$.
3: **for** $n = 1$ to $N$ **do**
4:   Compute the conditional cluster belief $\beta_{C_n|E}(\cdot, \mathbf{w}^{n-1}) \propto p_{C_n|E\setminus C_n}(\cdot \,|\, \mathbf{w}^{n-1}_{E\setminus C_n})$ via (9).
   *Advance the Markov chain:*
5:   Marginalize over $X_{C_n\setminus E}$ to obtain the transition density $p_{C_n\cap E|E\setminus C_n}(\cdot \,|\, \mathbf{w}^{n-1}_{E\setminus C_n})$.
6:   Sample $\mathbf{w}^n_{C_n\cap E} \sim p_{C_n\cap E|E\setminus C_n}(\cdot \,|\, \mathbf{w}^{n-1}_{E\setminus C_n})$ and set $\mathbf{w}^n_{E\setminus C_n} = \mathbf{w}^{n-1}_{E\setminus C_n}$.
   *Update any estimates to be computed at $C_n$:*
7:   **if** $D \subseteq C_n \cup E$ **then**
8:     Instantiate $\mathbf{w}^n_{C_n}$ in $\beta_{C^n|E}$ and normalize to obtain $p_{C_n\setminus E|E}(\cdot \,|\, \mathbf{w}^n)$.
9:     Compute the expectation $\mathbb{E}\left[f(X_D) \,|\, \mathbf{w}^n\right]$ via (7).
10:     Set $\hat{f} = \hat{f} + \mathbb{E}\left[f(X_D) \,|\, \mathbf{w}^n\right]$ and increment the sample count $M$.
   *Take the next step of the walk:*
11:   Choose a cluster $C_{n+1}$ that is a neighbor of $C_n$.
12:   Recompute the message $\mu_{C_n C_{n+1}|E}(\cdot, \mathbf{w}^n)$ via (8).
13: Set $\hat{f} \leftarrow \hat{f}/M$.

the freedom to later instantiate $X_{C\cap E}$ as we wish, or not at all. It is easy to verify that the *conditional belief* $\beta_{C|E}$ given by

$$\beta_{C|E}(\mathbf{u}, \mathbf{w}) \equiv \psi_C(\mathbf{u}) \prod_{(B,C)\in \mathbf{E}} \mu_{BC|E}(\mathbf{u}_B, \mathbf{w}), \qquad \mathbf{u} \in \mathbb{X}_C, \mathbf{w} \in \mathbb{X}_E \tag{9}$$

is proportional to the conditional density $p_{C|E\setminus C}(\mathbf{u} \,|\, \mathbf{w}_{E\setminus C})$.[3]

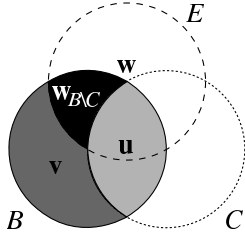

Figure 1: A Venn diagram showing how the ranges of the assignment variables in (8) cover the cluster $B$.

Using these modified definitions, we can dramatically reduce the number of messages computed per sample. In particular, the conditional messages have the following important property:

**Proposition.** Let $\mathbf{w}$ and $\mathbf{w}'$ be two assignments to $E$ such that $\mathbf{w}_{E\setminus D} = \mathbf{w}'_{E\setminus D}$ for some cluster $D$. Then for all edges $B \to C$ with $C$ closer to $D$ than $B$, $\mu_{BC|E}(\mathbf{u}, \mathbf{w}) = \mu_{BC|E}(\mathbf{u}, \mathbf{w}')$.

*Proof.* Assume by induction that the messages into $B$ (except the one from $C$) are equal given $\mathbf{w}$ or $\mathbf{w}'$. There are two cases to consider. If $(E \cap D)$ has no overlap with $(E \cap B)$, then $\mathbf{w}_{B\setminus C} = \mathbf{w}'_{B\setminus C}$ and the equality follows from (8). Otherwise, by the junction property we know that if $i \in B$ and $i \in D$, then $i \in C$, so again we get $\mathbf{w}_{B\setminus C} = \mathbf{w}'_{B\setminus C}$. $\qquad \square$

Thus, when we resample a cluster $C_n$, we have $\mathbf{w}^n_{E\setminus C_n} = \mathbf{w}^{n-1}_{E\setminus C_n}$ and so only those messages directed away from $C_n$ change. In addition, as argued above, when we resample $C_{n+1}$ in iteration $n + 1$, we only require the messages directed towards $C_{n+1}$. Combining these two arguments, we find that *only the messages on the directed path from $C_n$ to $C_{n+1}$ must be recomputed in iteration $n$*. If we choose $C_{n+1}$ to be a neighbor of $C_n$, we only have to recompute a single message in each iteration.[4] Putting all of these optimizations together, we obtain Algorithm 3, which is easily generalized to the case where many function expectations are computed in parallel.

### 3.3  Complexity of Sample Propagation

For simplicity of analysis we assume finite-value variables and tabular potentials. In the Shafer–Shenoy algorithm, the space complexity of representing the exact message (3) is $O(|\mathbb{X}_{B\cap C}|)$, and the time complexity of computing it is $O(|\mathbb{X}_B|)$ (since for each assignment to $B\cap C$ we must sum over assignments to $B\backslash C$). In contrast, when computing the conditional message (8), we only sum over assignments to $B\backslash(C\cup E)$, since $E\cap(B\backslash C)$ is instantiated by the current sample. This makes the conditional message cheaper to compute than the exact message: in the finite case the time complexity is $O(|\mathbb{X}_{B\backslash(E\cap(B\backslash C))}|)$. The space complexity of representing the conditional message is $O(|\mathbb{X}_{B\cap C}|)$—the same as the exact message, since it a potential over the same variables.

As we sample more variables, the conditional messages become cheaper to compute. However, note that the space complexity of representing the conditional message is independent of the choice of sampled variables $E$; even if we sample a given variable, it remains a free parameter of the conditional message. (If we instead fixed its value, the proposition above would not hold.) Thus, the time complexity of computing conditional messages can be reduced by sampling more variables, but only up to a point: the time complexity of computing the conditional message must be $o(|\mathbb{X}_{B\cap C}|)$. This contrasts with the approach of Bidyuk & Dechter [7], where the asymptotic time complexity of each iteration can be reduced arbitrarily by sampling more variables. However, to achieve this their algorithm runs the entire junction tree algorithm in each iteration, and does not reuse messages between iterations. In contrast, Sample Propagation reuses all but one of the messages between iterations, leading to a greatly reduced "constant factor".

## 4  Application to conditional Gaussian models

A *conditional Gaussian (CG) model* is a probability distribution over a set of discrete variables $\{X_i : i \in \Delta\}$ and continous variables $\{X_i : i \in \Gamma\}$ such that the conditional distribution of $X_\Gamma$ given $X_\Delta$ is multivariate Gaussian. Inference in CG models is harder than in models that are totally discrete or totally Gaussian. For example, consider polytree models: when all of the variables are discrete or all are Gaussian, exact inference is linear in size of the model; but if the model is CG then *approximate* inference is NP-hard [11].

In traditional junction tree inference, our goal is to compute the marginal for each cluster. However, when $p$ is a CG model, each cluster marginal is a mixture of $|\mathcal{X}_\Delta|$ Gaussians, and is intractable to represent. Instead, we can compute the *weak marginals*, i.e., for each cluster we compute the best conditional Gaussian approximation of $p_C$. Lauritzen's algorithm [12] is an extension of the HUGIN algorithm that computes these weak marginals exactly. Unfortunately, it is often intractable because it requires *strongly rooted* junction trees, which can have clusters that contain most or all of the discrete variables [3].

The structure of CG models makes it possible to use Sample Propagation to approximate the weak cluster marginals: we choose $E = \Delta$, since conditioning on the discrete variables leaves a tractable Gaussian inference problem.[5] The expectations we must compute are of the sufficient statistics of the weak cluster marginals: for each cluster $C$, we need the distribution of $X_{C\cap\Delta}$ and the conditional means and covariances of $X_{C\cap\Gamma}$ given $X_{C\cap\Delta}$.

As an example, consider the model given in Figure 2(a) for tracking an object whose state (position and velocity) at time $t$ is $X_t$. At each time step, we obtain a vector measurement $Y_t$ which is either a noisy measurement of the object's position (if $Z_t = 0$) or an outlier (if $Z_t = 1$). The Markov chain over $Z_t$ makes it likely that inliers and outliers come in bursts. The task is to estimate the position of the object at all time steps (for $T = 100$).

Lauritzen's algorithm is intractable in this case because any strongly rooted junction tree for this network must have a cluster containing all of the discrete variables [3, Thm. 3.18]. Therefore, instead of comparing our approximate position estimates to the correct answer, we sampled a trajectory from the network and computed the average position error to the (unobserved) ground truth. Both Gibbs sampling and Sample Propagation were run with a forwards–backwards sampling schedule; Sample Propagation used the junction tree of Figure 2(b).[6] Both algorithms were started in the same state and both were allowed to "burn in" for five forwards–backwards passes. We repeated this 10 times and averaged the results over trials. Figure 2(c) shows that Sample Propagation converged much more quickly than Gibbs sampling. Also, Sample Propagation found better answers than Assumed Density Filtering (a standard algorithm for this problem), but at increased computational cost.

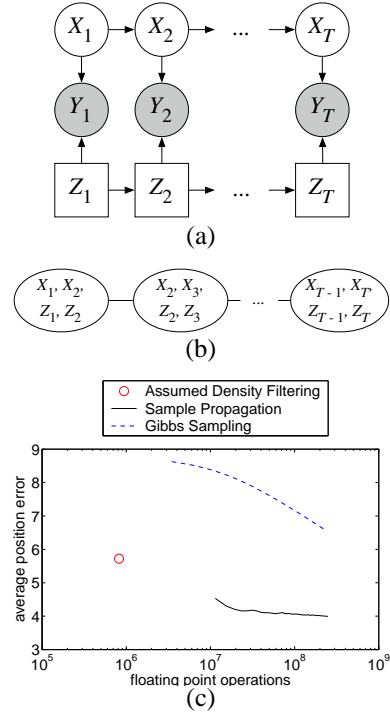

(a)

(b)

(c)

Figure 2: The TRACKING example.

**Acknowledgements.** I thank K. Murphy and S. Russell for comments on a draft of this paper. This research was supported by ONR N00014-00-1-0637 and an Intel Internship.

## Footnotes

[1]Interestingly, the blocking Gibbs proposal [9] makes a different use of junction tree inference than we do here: they use message passing *within* a block of variables to efficiently generate a sample.

[2]In cases where the transition density $p_{C \cap E|E \setminus C}(\cdot \mid \mathbf{w}_{E \setminus C}^n)$ is too large to represent or too difficult to sample from, we can use the Metropolis-Hastings algorithm, where we instead sample from a simpler proposal distribution $q_{C \cap E}$ and then accept or reject the proposal [8].

[3]The modified message passing scheme we describe can be viewed as an implementation of *fast retraction* for Shafer-Shenoy messages, analogous to the scheme described for HUGIN in [2, §6.4.6].

[4]A similar idea has recently been used to improve the efficiency of the Unified Propagation and Scaling algorithm for maximum likelihood estimation [10].

[5]We cannot choose $E = \Gamma$ because computing the conditional messages (8) may require summing discrete variables out of CG potentials, which leads to representational difficulties [3]. In this case one can instead use Bidyuk & Dechter's algorithm, which does not require these operations.

[6]Carter & Kohn [13] describe a specialized algorithm for this model that is similar to a version of Sample Propagation that does not resample the discrete variables on the backwards pass.

# References

[1] G. Shafer and P. Shenoy. Probability propagation. *Annals of Mathematics and Artificial Intelligence*, 2:327–352, 1990.

[2] R. Cowell, P. Dawid, S. Lauritzen, and D. Spiegelhalter. *Probabilistic Networks and Expert Systems*. Springer, 1999.

[3] U. Lerner. *Hybrid Bayesian Networks for Reasoning About Complex Systems*. PhD thesis, Stanford University, October 2002.

[4] A. Dawid, U. Kjærulff, and S. Lauritzen. Hybrid propagation in junction trees. In *Advances in Intelligent Computing*, volume 945 of *Lecture Notes in Computer Science*. Springer, 1995.

[5] U. Kjærulff. HUGS: Combining exact inference and Gibbs sampling in junction trees. In *Proc. of the 11th Conf. on Uncertainty in Artificial Intelligence (UAI-95)*. Morgan Kaufmann, 1995.

[6] A. Doucet, N. de Freitas, K. Murphy, and S. Russell. Rao-Blackwellised particle filtering for dynamic Bayesian networks. In *Proc. of the 16th Conf. on Uncertainty in AI (UAI-00)*, 2000.

[7] B. Bidyuk and R. Dechter. An empirical study of w-cutset sampling for Bayesian networks. In *Proc. of the 19th Conf. on Uncertainty in AI (UAI-03)*. Morgan Kaufmann, 2003.

[8] R. Neal. Probabilistic inference using Markov chain Monte Carlo methods. Technical Report CRG-TR-93-1, University of Toronto, 1993.

[9] C. S. Jensen, A. Kong, and U. Kjærulff. Blocking Gibbs sampling in very large probabilistic expert systems. *International Journal of Human-Computer Studies*, 42:647–666, 1995.

[10] Y. W. Teh and M. Welling. On improving the efficiency of the iterative proportional fitting procedure. In *Proc. of the 9th Int'l. Workshop on AI and Statistics (AISTATS-03)*, 2003.

[11] U. Lerner and R. Parr. Inference in hybrid networks: Theoretical limits and practical algorithms. In *Proc. of the 17th Conf. on Uncertainty in AI (UAI-01)*. Morgan Kaufmann, 2001.

[12] S. Lauritzen. Propagation of probabilities, means, and variances in mixed graphical association models. *Journal of the American Statistical Association*, 87(420):1098–1108, 1992.

[13] C. Carter and R. Kohn. Markov chain Monte Carlo in conditionally Gaussian state space models. *Biometrika*, 83:589–601, 1996.

